# Discriminative Learning of Sum-Product Networks

**Robert Gens**         **Pedro Domingos**
Department of Computer Science and Engineering
University of Washington
Seattle, WA 98195-2350, U.S.A.
{rcg,pedrod}@cs.washington.edu

## Abstract

Sum-product networks are a new deep architecture that can perform fast, exact inference on high-treewidth models. Only generative methods for training SPNs have been proposed to date. In this paper, we present the first discriminative training algorithms for SPNs, combining the high accuracy of the former with the representational power and tractability of the latter. We show that the class of tractable discriminative SPNs is broader than the class of tractable generative ones, and propose an efficient backpropagation-style algorithm for computing the gradient of the conditional log likelihood. Standard gradient descent suffers from the diffusion problem, but networks with many layers can be learned reliably using "hard" gradient descent, where marginal inference is replaced by MPE inference (i.e., inferring the most probable state of the non-evidence variables). The resulting updates have a simple and intuitive form. We test discriminative SPNs on standard image classification tasks. We obtain the best results to date on the CIFAR-10 dataset, using fewer features than prior methods with an SPN architecture that learns local image structure discriminatively. We also report the highest published test accuracy on STL-10 even though we only use the labeled portion of the dataset.

## 1   Introduction

Probabilistic models play a crucial role in many scientific disciplines and real world applications. Graphical models compactly represent the joint distribution of a set of variables as a product of factors normalized by the partition function. Unfortunately, inference in graphical models is generally intractable. Low treewidth ensures tractability, but is a very restrictive condition, particularly since the highest practical treewidth is usually 2 or 3 [2, 9]. Sum-product networks (SPNs) [23] overcome this by exploiting context-specific independence [7] and determinism [8]. They can be viewed as a new type of deep architecture, where sum layers alternate with product layers. Deep networks have many layers of hidden variables, which greatly increases their representational power, but inference with even a single layer is generally intractable, and adding layers compounds the problem [3]. SPNs are a deep architecture with full probabilistic semantics where inference is guaranteed to be tractable, under general conditions derived by Poon and Domingos [23]. Despite their tractability, SPNs are quite expressive [16], and have been used to solve difficult problems in vision [23, 1].

Poon and Domingos introduced an algorithm for generatively training SPNs, yet it is generally observed that discriminative training fares better. By optimizing $P(\mathbf{Y}|\mathbf{X})$ instead of $P(\mathbf{X}, \mathbf{Y})$ conditional random fields retain joint inference over dependent label variables $\mathbf{Y}$ while allowing for flexible features over given inputs $\mathbf{X}$ [22]. Unfortunately, the conditional partition function $Z(\mathbf{X})$ is just as prone to intractability as with generative training. For this reason, low treewidth models (e.g. chains and trees) of $\mathbf{Y}$ are commonly used. Research suggests that approximate inference can make it harder to learn rich structured models [21]. In this paper, discriminatively training SPNs will allow us to combine flexible features with fast, exact inference over high treewidth models.

With inference and learning that easily scales to many layers, SPNs can be viewed as a type of deep network. Existing deep networks employ discriminative training with backpropagation through softmax layers or support vector machines over network variables. Most networks that are not purely feed-forward require approximate inference. Poon and Domingos showed that deep SPNs could be learned faster and more accurately than deep belief networks and deep Boltzmann machines on a generative image completion task [23]. This paper contributes a discriminative training algorithm that could be used on its own or with generative pre-training.

For the first time we combine the advantages of SPNs with those of discriminative models. In this paper we will review SPNs and describe the conditions under which an SPN can represent the conditional partition function. We then provide a training algorithm, demonstrate how to compute the gradient of the conditional log-likelihood of an SPN using backpropagation, and explore variations of inference. Finally, we show state-of-the-art results where a discriminatively-trained SPN achieves higher accuracy than SVMs and deep models on image classification tasks.

## 2    Sum-Product Networks

SPNs were introduced with the aim of identifying the most expressive tractable representation possible. The foundation for their work lies in Darwiche's network polynomial [14]. We define an unnormalized probability distribution $\Phi(\mathbf{x}) \geq 0$ over a vector of Boolean variables $\mathbf{X}$. The indicator function $[.]$ is one when its argument is true and zero otherwise; we abbreviate $[X_i]$ and $[\bar{X}_i]$ as $x_i$ and $\bar{x}_i$. To distinguish random variables from indicator variables, we use roman font for the former and italic for the latter. Vectors of variables are denoted by bold roman and bold italic font, respectively. The network polynomial of $\Phi(\mathbf{x})$ is defined as $\sum_{\mathbf{x}} \Phi(\mathbf{x}) \prod(\mathbf{x})$, where $\prod(\mathbf{x})$ is the product of indicators that are one in state $\mathbf{x}$. For example, the network polynomial of the Bayesian network $X_1 \to X_2$ is $P(x_1)P(x_2|x_1)x_1x_2 + P(x_1)P(\bar{x}_2|x_1)x_1\bar{x}_2 + P(\bar{x}_1)P(x_2|\bar{x}_1)\bar{x}_1x_2 + P(\bar{x}_1)P(\bar{x}_2|\bar{x}_1)\bar{x}_1\bar{x}_2$. To compute $P(X_1 = \text{true}, X_2 = \text{false})$, we access the corresponding term of the network polynomial by setting indicators $x_1$ and $\bar{x}_2$ to one and the rest to zero. To find $P(X_2 = \text{true})$, we fix evidence on $X_2$ by setting $x_2$ to one and $\bar{x}_2$ to zero and marginalize $X_1$ by setting both $x_1$ and $\bar{x}_1$ to one. Notice that there are two reasons we might set an indicator $x_i = 1$: *(1)* evidence $\{X_i = \text{true}\}$, in which case we set $\bar{x}_i = 0$ and *(2)* marginalization of $X_i$, where $\bar{x}_i = 1$ as well. In general the role of an indicator $x_i$ is to determine whether terms compatible with variable state $X_i = \text{true}$ are included in the summation, and similarly for $\bar{x}_i$. With this notation, the partition function $Z$ can be computed by setting all indicators of all variables to one.

The network polynomial has size exponential in the number of variables, but in many cases it can be represented more compactly using a sum-product network [23, 14].

**Definition 1.** *(Poon & Domingos, 2011) A sum-product network (SPN) over variables $X_1, \ldots, X_d$ is a rooted directed acyclic graph whose leaves are the indicators $x_1, \ldots, x_d$ and $\bar{x}_1, \ldots, \bar{x}_d$ and whose internal nodes are sums and products. Each edge $(i, j)$ emanating from a sum node $i$ has a non-negative weight $w_{ij}$. The value of a product node is the product of the values of its children. The value of a sum node is $\sum_{j \in Ch(i)} w_{ij} v_j$, where $Ch(i)$ are the children of $i$ and $v_j$ is the value of node $j$. The value of an SPN $S[x_1, \bar{x}_1, \ldots, x_d, \bar{x}_d]$ is the value of its root.*

If we could replace the exponential sum over variable states in the partition function with the linear evaluation of the network, inference would be tractable. For example, the SPN in Figure 1 represents the joint probability of three Boolean variables $P(X_1, X_2, X_3)$ in the Bayesian network $X_2 \leftarrow X_1 \to X_3$ using six indicators $S[x_1, \bar{x}_1, x_2, \bar{x}_2, x_3, \bar{x}_3]$. To compute $P(X_1 = \text{true})$, we could sum over the joint states of $X_2$ and $X_3$, evaluating the network a total of four times $S[1, 0, 0, 1, 0, 1] + \ldots + S[1, 0, 1, 0, 1, 0]$. Instead, we set the indicators so that the network sums out both $X_2$ and $X_3$. An indicator setting of S[1,0,1,1,1,1] computes

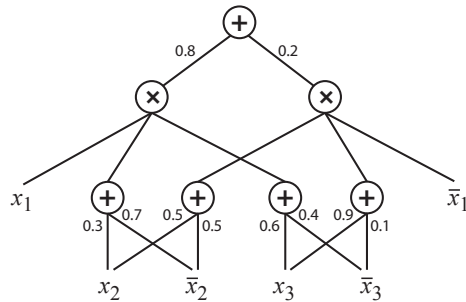

Figure 1: SPN over Boolean variables $X_1, X_2, X_3$

the sum over all states compatible with our evidence e = {X$_1$ = true} and requires only one evaluation.

However, not every SPN will have this property. If a linear evaluation of an SPN with indicators set to represent evidence equals the exponential sum over all variable states consistent with that evidence, the SPN is *valid*.

**Definition 2.** *(Poon & Domingos, 2011) A sum-product network S is* valid *iff* $S(e) = \Phi_S(e)$ *for all evidence* e.

In their paper, Poon and Domingos prove that there are two conditions sufficient for validity: completeness and consistency.

**Definition 3.** *(Poon & Domingos, 2011) A sum-product network is* complete *iff all children of the same sum node have the same scope.*

**Definition 4.** *(Poon & Domingos, 2011) A sum-product network is* consistent *iff no variable appears negated in one child of a product node and non-negated in another.*

**Theorem 1.** *(Poon & Domingos, 2011) A sum-product network is valid if it is complete and consistent.*

The scope of a node is defined as the set of variables that have indicators among the node's descendants. To "appear in a child" means to be among that child's descendants. If a sum node is incomplete, the SPN will *undercount* the true marginals. Since an incomplete sum node has scope larger than a child, that child will be non-zero for more than one state of the sum (e.g. if $S[x_1, \bar{x}_1, x_2, \bar{x}_2] = (x_1 + x_2)$, $S[1, 0, 1, 1] < S[1, 0, 1, 0] + S[1, 0, 0, 1]$). If a product node is inconsistent, the SPN will *overcount* the marginals as it will incorporate impossible states (e.g. $x_1 \times \bar{x}_1$) into its computation.

Poon and Domingos show how to generatively train the parameters of an SPN. One method is to compute the likelihood gradient and optimize with gradient descent (GD). They also show how to use expectation maximization (EM) by considering each sum node as the marginalization of a hidden variable [17]. They found that online EM using most probable explanation (MPE or "hard") inference worked the best for their image completion task.

Gradient diffusion is a key issue in training deep models. It is commonly observed in neural networks that when the gradient is propagated to lower layers it becomes less informative [3]. When every node in the network takes fractional responsibility for the errors of a top level node, it becomes difficult to steer parameters out of local minima. Poon and Domingos also saw this effect when using gradient descent and EM to train SPNs. They found that online hard EM could provide a sparse but strong learning signal to synchronize the efforts of upper and lower nodes. Note that hard training is not exclusive to EM. In the next section we show how to discriminatively train SPNs with hard gradient descent.

## 3 Discriminative Learning of SPNs

We define an SPN $S[\boldsymbol{y}, \boldsymbol{h}|\boldsymbol{x}]$ that takes as input three disjoint sets of variables $\mathbf{H}$, $\mathbf{Y}$, and $\mathbf{X}$ (hidden, query, and given). We denote the setting of all $\boldsymbol{h}$ indicator functions to 1 as $S[\boldsymbol{y}, \boldsymbol{1}|\boldsymbol{x}]$, where the bold $\boldsymbol{1}$ is a vector. We do not sum over states of given variables $\mathbf{X}$ when discriminatively training SPNs. Given an instance, we treat $\mathbf{X}$ as constants. This means that one ignores $\mathbf{X}$ variables in the scope of a node when considering completeness and consistency. Since adding a constant as a child to a product node cannot make that product inconsistent, a variable $x$ can be the child of any product node in a valid SPN. To maintain completeness, $x$ can only be the child of a sum node that has scope outside of $\mathbf{Y}$ or $\mathbf{H}$.

---

**Algorithm 1:** LearnSPN

**Input**: Set $D$ of instances over variables $\mathbf{X}$ and label variables $\mathbf{Y}$, a valid SPN $S$ with initialized parameters.
**Output**: An SPN with learned weights
**repeat**
    **forall the** $d \in D$ **do**
        └ UpdateWeights(S, Inference($S$,$\mathbf{x_d}$,$\mathbf{y_d}$))
**until** *convergence or early stopping condition*;

---

The parameters of an SPN can be learned using an online procedure as in Algorithm 1 as proposed by Poon and Domingos. The three dimensions of the algorithm are generative vs. discriminative, the inference procedure, and the weight update. Poon and Domingos discussed generative gradient descent with marginal inference as well as EM with marginal and MPE inference. In this section we will derive discriminative gradient descent with marginal and MPE inference, where hard gradient descent can also be used for generative training. EM is not typically used for discriminative training as it requires modification to lower bound the conditional likelihood [25] and there may not be a closed form for the M-step.

## 3.1 Discriminative Training with Marginal Inference

A component of the gradient of the conditional log likelihood takes the form

$$
\begin{aligned}
\frac{\partial}{\partial w} \log P(\mathbf{y}|\mathbf{x}) &= \frac{\partial}{\partial w} \log \sum_{\mathbf{h}} \Phi(\mathbf{Y} = \mathbf{y}, \mathbf{H} = \mathbf{h}|\mathbf{x}) - \frac{\partial}{\partial w} \log \sum_{\mathbf{y}',\mathbf{h}} \Phi(\mathbf{Y} = \mathbf{y}', \mathbf{H} = \mathbf{h}|\mathbf{x}) \\
&= \frac{1}{S[\boldsymbol{y}, \mathbf{1}|\boldsymbol{x}]} \frac{\partial S[\boldsymbol{y}, \mathbf{1}|\boldsymbol{x}]}{\partial w} - \frac{1}{S[\mathbf{1}, \mathbf{1}|\boldsymbol{x}]} \frac{\partial S[\mathbf{1}, \mathbf{1}|\boldsymbol{x}]}{\partial w}
\end{aligned}
$$

where the two summations are separate bottom-up evaluations of the SPN with indicators set as $S[\boldsymbol{y}, \mathbf{1}|\boldsymbol{x}]$ and $S[\mathbf{1}, \mathbf{1}|\boldsymbol{x}]$, respectively.

The partial derivatives of the SPN with respect to all weights can be computed with backpropagation, detailed in Algorithm 2. After performing a bottom-up evaluation of the SPN, partial derivatives are passed from parent to child as follows from the chain rule and described in [15]. The form of backpropagation presented takes time linear in the number of nodes in the SPN if product nodes have a bounded number of children.

Our gradient descent update then follows the direction of the partial derivative of the conditional log likelihood with learning rate $\eta$: $\Delta w = \eta \frac{\partial}{\partial w} \log P(\mathbf{y}|\mathbf{x})$. After each gradient step we optionally renormalize the weights of a sum node so they sum to one. Empirically we have found this to produce the best results. The second SPN evaluation that marginalizes $\mathbf{H}$ and $\mathbf{Y}$ can reuse computation from the first, for example, when $\mathbf{Y}$ is modeled by a root sum node. In this case the values of all non-root nodes are equivalent between the two evaluations. For any architecture, one can memoize values of nodes that do not have a query variable indicator as a descendant.

---

**Algorithm 2:** BackpropSPN

---

**Input**: A valid SPN $S$, where $S_n$ denotes the value of node $n$ after bottom-up evaluation.
**Output**: Partial derivatives of the SPN with respect to every node $\frac{\partial S}{\partial S_n}$ and weight $\frac{\partial S}{\partial w_{i,j}}$

Initialize all $\frac{\partial S}{\partial S_n} = 0$ except $\frac{\partial S}{\partial S} = 1$
**forall the** $n \in S$ *in top-down order* **do**
    **if** *n is a sum node* **then**
        **forall the** $j \in Ch(n)$ **do**
            $\frac{\partial S}{\partial S_j} \leftarrow \frac{\partial S}{\partial S_j} + w_{n,j} \frac{\partial S}{\partial S_n}$
            $\frac{\partial S}{\partial w_{n,j}} \leftarrow S_j \frac{\partial S}{\partial S_n}$
    **else**
        **forall the** $j \in Ch(n)$ **do**
            $\frac{\partial S}{\partial S_j} \leftarrow \frac{\partial S}{\partial S_j} + \frac{\partial S}{\partial S_n} \prod_{k \in Ch(n) \setminus \{j\}} S_k$

---

## 3.2 Discriminative Training with MPE Inference

There are several reasons why MPE inference is appealing for discriminatively training SPNs. As discussed above, hard inference was crucial for overcoming gradient diffusion when generatively training SPNs. For many applications the goal is to predict the most probable structure, and therefore it makes sense to use this also during training. Finally, it is common to approximate summations with maximizations for reasons of speed or tractability. Though summation in SPNs is fast and exact, MPE inference is still faster. We derive discriminative gradient descent using MPE inference.

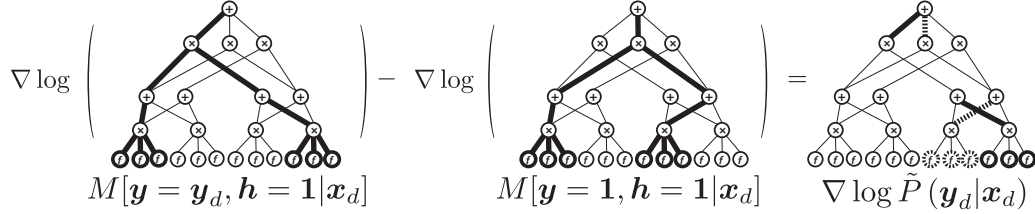

$$M[\boldsymbol{y} = \boldsymbol{y}_d, \boldsymbol{h} = \mathbf{1}|\boldsymbol{x}_d] \qquad M[\boldsymbol{y} = \mathbf{1}, \boldsymbol{h} = \mathbf{1}|\boldsymbol{x}_d] \qquad \nabla \log \tilde{P}\left(\boldsymbol{y}_d|\boldsymbol{x}_d\right)$$

Figure 2: Positive and negative terms in the hard gradient. The root node sums out the variable Y, the two sum nodes on the left sum out the hidden variable $H_1$, the two sum nodes on the right sum out $H_2$, and a circled 'f' denotes an input variable $X_i$. Dashed lines indicate negative elements in the gradient.

We define a max-product network (MPN) $M[\boldsymbol{y}, \boldsymbol{h}|\boldsymbol{x}]$ based on the max-product semiring. This network compactly represents the maximizer polynomial $\max_{\mathbf{x}} \Phi(\mathbf{x}) \prod(\mathbf{x})$, which computes the MPE [15]. To convert an SPN to an MPN, we replace each sum node by a max node, where weights on children are retained. The gradient of the conditional log likelihood with MPE inference is then

$$\frac{\partial}{\partial w} \log \tilde{P}(\mathbf{y}|\mathbf{x}) \;\; = \;\; \frac{\partial}{\partial w} \log \max_{\mathbf{h}} \Phi(\mathbf{Y} = \mathbf{y}, \mathbf{H} = \mathbf{h}|\mathbf{x}) - \frac{\partial}{\partial w} \log \max_{\mathbf{y}', \mathbf{h}} \Phi(\mathbf{Y} = \mathbf{y}', \mathbf{H} = \mathbf{h}|\mathbf{x})$$

where the two maximizations are computed by $M[\boldsymbol{y}, \mathbf{1}|\boldsymbol{x}]$ and $M[\mathbf{1}, \mathbf{1}|\boldsymbol{x}]$. MPE inference also consists of a bottom-up evaluation followed by a top-down pass. Inference yields a branching path through the SPN called a complete subcircuit that includes an indicator (and therefore assignment) for every variable [15]. Analogous to Viterbi decoding, the path starts at the root node and at each max (formerly sum) node it only travels to the max-valued child. At product nodes, the path branches to all children. We define $W$ as the multiset of weights traversed by this path[1]. The value of the MPN takes the form of a product $\prod_{w_i \in W} w_i^{c_i}$, where $c_i$ is the number of times $w_i$ appears in $W$. The partial derivatives of the MPN with respect to all nodes and weights is computed by Algorithm 2 modified to accommodate MPNs: *(1)* $S$ becomes $M$, *(2)* when $n$ is a sum node, the body of the *forall* loop is run once for $j$ as the max-valued child.

The partial derivative of the logarithm of an MPN with respect to a weight takes the form

$$\frac{\partial \log M}{\partial w_i} = \frac{\partial \log M}{\partial M} \frac{\partial M}{\partial w_i} = \frac{1}{M} \frac{\partial M}{\partial w_i} = \frac{c_i \cdot w_i^{c_i - 1} \prod_{w_j \in W \setminus \{w_i\}} w_j^{c_j}}{\prod_{w_j \in W} w_j^{c_j}} \;\; = \;\; \frac{c_i}{w_i}$$

The gradient of the conditional log likelihood with MPE inference is therefore $\Delta c_i / w_i$, where $\Delta c_i = c_i' - c_i''$ is the difference between the number of times $w_i$ is traversed by the two MPE inference paths in $M[\boldsymbol{y}, \mathbf{1}|\boldsymbol{x}]$ and $M[\mathbf{1}, \mathbf{1}|\boldsymbol{x}]$, respectively. The hard gradient update is then $\Delta w_i = \eta \frac{\partial}{\partial w_i} \log \tilde{P}(\mathbf{y}|\mathbf{x}) = \eta \frac{\Delta c_i}{w_i}$.

The hard gradient for a training instance $(\mathbf{x_d}, \mathbf{y_d})$ is illustrated in Figure 2. In the first two expressions, the complete subcircuit traveled by each MPE inference is shown in bold. Product nodes do not have weighted children, so they do not appear in the gradient, depicted in the last expression

We can also easily add regularization to SPN training. An L2 weight penalty takes the familiar form of $-\lambda ||\mathbf{w}||^2$ and partial derivatives $-2\lambda w_i$ can be added to the gradient. With an appropriate optimization method, an L1 penalty could also be used for learning with marginal inference on dense SPN architectures. However, sparsity is not as important for SPNs as it is for Markov random fields, where a non-zero weight can have outsize impact on inference time; with SPNs inference is always linear with respect to model size.

A summary of the variations of Algorithm 1 is provided in Tables 1 and 2. The generative hard gradient can be used in place of online EM for datasets where it would be prohibitive to store inference results from past epoch. For architectures that have high fan-in sum nodes, soft inference may be able to separate groups of modes faster than hard inference, which can only alter one child of a sum node at a time.

We observe the similarity between the updates of hard EM and hard gradient descent. In particular, if we reparameterize the SPN so that each child of a sum node is weighted by $w_i = e^{w_i'}$, the form of

Table 1: Inference procedures

| Node | Soft Inference | Hard Inference |
|---|---|---|
| Sum | $\frac{\partial S}{\partial S_n} = \sum_{k \in Pa(n)} \frac{\partial S}{\partial S_k} \prod_{l \in Ch(k)\backslash\{n\}} S_l$ | $\frac{\partial M}{\partial M_n} = \sum_{k \in Pa(n)} \frac{\partial M}{\partial M_k} \prod_{l \in Ch(k)\backslash\{n\}} M_l$ |
| Product | $\frac{\partial S}{\partial S_n} = \sum_{k \in Pa(n)} w_{kn} \frac{\partial S}{\partial S_k}$ | $\frac{\partial M}{\partial M_n} = \sum_{k \in Pa(n)} \begin{cases} w_{kn} \frac{\partial M}{\partial M_k} & : w_{kn} \in W \\ 0 & : \text{otherwise} \end{cases}$ |
| Weight | $\frac{\partial S}{\partial w_{ki}} = \frac{\partial S}{\partial S_k} S_i$ | $\frac{\partial M}{\partial w_{ki}} = \frac{\partial M}{\partial M_k} M_i$ |

Table 2: Weight updates

| Update | Soft Inference | Hard Inference |
|---|---|---|
| Gen. GD | $\Delta w = \eta \frac{\partial S[\boldsymbol{x},\boldsymbol{y}]}{\partial w}$ | $\Delta w_i = \eta \frac{c_i}{w_i}$ |
| Gen. EM | $P(\mathrm{H_k} = \mathrm{i}|\mathbf{x}, \mathbf{y}) \propto w_{ki} \frac{\partial S[\boldsymbol{x},\boldsymbol{y}]}{\partial S_k}$ | $P(\mathrm{H_k} = \mathrm{i}|\mathbf{x}, \mathbf{y}) = \begin{cases} 1 & : w_{ki} \in W \\ 0 & : \text{otherwise} \end{cases}$ |
| Disc. GD | $\Delta w = \eta \left( \frac{1}{S[\boldsymbol{y},\mathbf{1}|\boldsymbol{x}]} \frac{\partial S[\boldsymbol{y},\mathbf{1}|\boldsymbol{x}]}{\partial w} - \frac{1}{S[\mathbf{1},\mathbf{1}|\boldsymbol{x}]} \frac{\partial S[\mathbf{1},\mathbf{1}|\boldsymbol{x}]}{\partial w} \right)$ | $\Delta w_i = \eta \frac{\Delta c_i}{w_i}$ |

the partial derivative of the log MPN becomes

$$\frac{\partial \log M}{\partial w_i'} = \frac{1}{M} \frac{\partial M}{\partial w_i'} = \frac{c_i \prod_{w_j' \in W'} e^{c_j \cdot w_j'}}{\prod_{w_j' \in W'} e^{c_j \cdot w_j'}} = c_i$$

This means that the hard gradient update for weights in logspace is $\Delta w_i' = \Delta c_i$, which resembles structured perceptron [13].

## 4 Experiments

We have applied discriminative training of SPNs to image classification benchmarks. CIFAR-10 and STL-10 are standard datasets for deep networks and unsupervised feature learning. Both are 10-class small image datasets. We achieve the best results to date on both tasks.

We follow the feature extraction pipeline of Coates et al. [10], which was also used recently to learn pooling functions [20]. The procedure consists of extracting $4 \times 10^5$ 6x6 pixel patches from the training set images, ZCA whitening those patches [19], running k-means for 50 rounds, and then normalizing the dictionary to have zero mean and unit variance. We then use the dictionary to extract $K$ features at every 6x6 pixel site in the image (unit stride) with the "triangle" encoding $f_k(x) = \max\{0, \bar{z} - z_k\}$, where $z_k = ||x - c_k||_2$, $c_k$ is the k-th item in the dictionary, and $\bar{z}$ is the average $z_k$. For each image of CIFAR-10, for example, this yields a $27 \times 27 \times K$ feature vector that is finally downsampled by max-pooling to a $G \times G \times K$ feature vector.

We experiment with a simple architecture that allows for discriminative learning of local structure. This architecture cannot be generatively trained as it violates consistency over $\mathbf{X}$. Inspired by the successful star models in Felzenszwalb et al. [18], we construct a network with $C$ classes, $P$ parts per class, and $T$ mixture components per part. A part is a pattern of image patch features that can occur anywhere in the image (e.g. an arrangement of patches that defines a curve). Each part filter $\vec{f}_{cpt}$ is of dimension $W \times W \times K$ and is initialized to $\vec{0}$. The root of the SPN is a sum node with a child $S_c$ for each class $c$ in the dataset multiplied by the indicator for that state of the label variable Y. $S_c$ is a product over $P$ nodes $S_{cp}$, where each $S_{cp}$ is a sum node over $T$ nodes

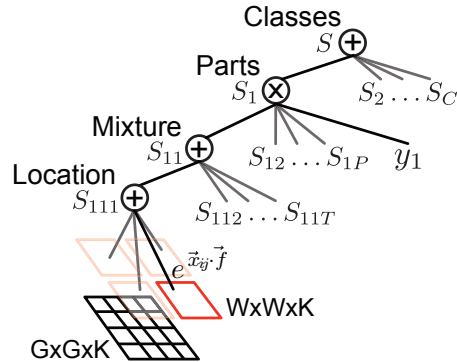

Figure 3: SPN architecture for experiments. Hidden variable indicators omitted for legibility.

$S_{cpt}$. The hidden variables **H** represent the choice of cluster in the mixture over a part and its position ($S_{cp}$ and $S_{cpt}$, respectively). Finally, $S_{cpt}$ sums over positions $i, j$ in the image of the logistic function $e^{\vec{x}_{ij} \cdot \vec{f}_{cpt}}$ where the given variable $\vec{x}_{ij}$ is the same dimension as $f$ and parts can overlap.

Notice that the mixture $S_{cp}$ models an additional level of spatial structure on top of the image patch features learned by k-means. Coates and Ng [12] also learn higher-order structure, but whereas our method learns structure discriminatively in the context of a parts-based model, their unsupervised algorithm greedily groups features based on correlation and is unable to learn mixtures. Compared with the pooling functions in Jia et al. [20] that model independent translation of patch features, our architecture models how nearby features move together. Other deep probabilistic architectures should be able to model high-level structure, but considering the difficulty in training these models with approximate inference, it is hard to make full use of their representational power. Unlike the star model of Felzenswalb et al. [18] that learns filters over predefined HOG image features, our SPN learns on top of learned image features that can model color and detailed patterns.

Generative SPN architectures on the same features produce unsatisfactory results as generative training is led astray by the large number of features, very few of which differentiate labels. In the generative SPN paper [23], continuous variables are modeled with univariate Gaussians at the leaves (viewed as a sum node with infinite children but finite weight sum). With discriminative training, **X** can be continuous because we always condition on it, which effectively folds it into the weights.

All networks are learned with stochastic gradient descent regularized by early stopping. We found that using marginal inference for the root node and MPE inference for the rest of the network worked best. This allows the SPN to continue learning the difference between classes even when it correctly classifies a training instance. The fraction of the training set reserved for validation with CIFAR-10 and STL-10 were 10% and 20%, respectively. Learning rates, $P$, and $T$ were chosen based on validation set performance.

## 4.1 Results on CIFAR-10

CIFAR-10 consists of 32x32 pixel images: $5 \times 10^4$ for training and $10^4$ for testing. We first compare discriminative SPNs with other methods as we vary the size of the dictionary $K$. The results are seen in Figure 4. To fairly compare with recent work [10, 20] we also set $G = 4$. In general, we observe that SPNs can achieve higher performance using half as many features as the next best approach, the learned pooling function. We hypothesize that this is because the SPN architecture allows us to discriminatively train large moveable parts, image structure that cannot be captured by larger dictionaries. In Jia et al. [20] the pooling functions blur individual features (i.e. a 6x6 pixel dictionary item), from which the classifier may have trouble inferring the coordination of image parts.

We then experimented with a finer grid and fewer dictionary items ($G = 7$, $K = 400$). Pooling functions destroy information, so it is better if less is done before learning. Finer grids are less feasible for the method in Jia et al. [20] as the number of rectangular pooling functions grows $O(G^4)$. Our best test accuracy of 83.96% was achieved with $W = 3$, $P = 200$, and $T = 2$, chosen

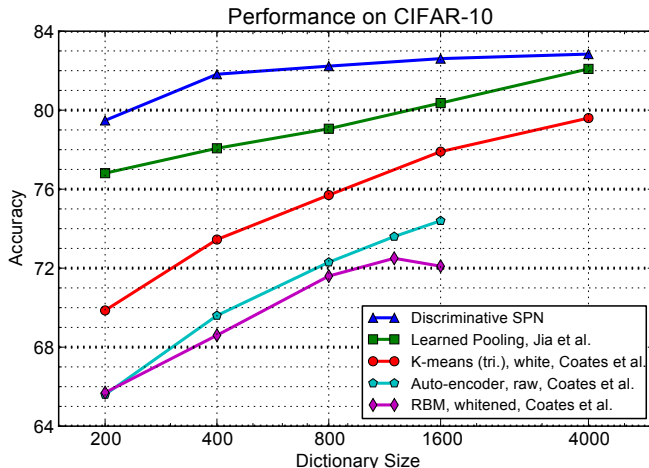

Figure 4: Impact of dictionary size $K$ with a 4x4 pooling grid ($W=3$) on CIFAR-10 test accuracy

Table 3: Test accuracies on CIFAR-10.

| Method | Dictionary | Accuracy |
|---|---|---|
| Logistic Regression [24] | | 36.0% |
| SVM [5] | | 39.5% |
| SIFT [5] | | 65.6% |
| mcRBM [24] | | 68.3% |
| mcRBM-DBN [24] | | 71.0% |
| Convolutional RBM [10] | | 78.9% |
| K-means (Triangle) [10] | 4000, 4x4 grid | 79.6 % |
| HKDES [4] | | 80.0% |
| 3-Layer Learned RF [12] | 1600, 9x9 grid | 82.0% |
| Learned Pooling [20] | 6000, 4x4 grid | 83.11% |
| **Discriminative SPN** | **400, 7x7 grid** | **83.96%** |

Table 4: Comparison of average test accuracies on all folds of STL-10.

| Method | Accuracy ($\pm\sigma$) |
|---|---|
| 1-layer Vector Quantization [11] | 54.9% ($\pm$ 0.4%) |
| 1-layer Sparse Coding [11] | 59.0% ($\pm$ 0.8%) |
| 3-layer Learned Receptive Field [12] | 60.1% ($\pm$ 1.0%) |
| **Discriminative SPN** | **62.3% ($\pm$ 1.0%)** |

by validation set performance. This architecture achieves the highest published test accuracy on the CIFAR-10 dataset, remarkably using one fifth the number of features of the next best approach. We compare top CIFAR-10 results in Table 3, highlighting the dictionary size of systems that use the feature extraction from Coates et al. [10].

## 4.2 Results on STL-10

STL-10 has larger 96x96 pixel images and less labeled data (5,000 training and 8,000 test) than CIFAR-10 [10]. The training set is mapped to ten predefined folds of 1,000 images. We experimented on the STL-10 dataset in a manner similar to CIFAR-10, ignoring the $10^5$ items of unlabeled data. Ten models were trained on the pre-specified folds, and test accuracy is reported as an average. With $K=1600$, $G=8$, $W=4$, $P=10$, and $T=3$ we achieved 62.3% ($\pm$ 1.0% standard deviation among folds), the highest published test accuracy as of writing. Notably, this includes approaches that make use of the unlabeled training images. Like Coates and Ng [12], our architecture learns local relations among different feature maps. However, the SPN is able to discriminatively learn latent mixtures, which can encode a more nuanced decision boundary than the linear classifier used in their work. After we carried out our experiments, Bo et al. [6] reported a higher accuracy with their unsupervised features and a linear SVM. Just as with the features of Coates et al. [10], we anticipate that using an SPN instead of the SVM would be beneficial by learning spatial structure that the SVM cannot model.

## 5 Conclusion

Sum-product networks are a new class of probabilistic model where inference remains tractable despite high treewidth and many hidden layers. This paper introduced the first algorithms for learning SPNs discriminatively, using a form of backpropagation to compute gradients. Discriminative training allows for a wider variety of SPN architectures than generative training, because completeness and consistency do not have to be maintained over evidence variables. We proposed both "soft" and "hard" gradient algorithms, using marginal inference in the "soft" case and MPE inference in the "hard" case. The latter successfully combats the diffusion problem, allowing deep networks to be learned. Experiments on image classification benchmarks illustrate the power of discriminative SPNs.

Future research directions include applying other discriminative learning paradigms to SPNs (e.g. max-margin methods), automatically learning SPN structure, and applying discriminative SPNs to a variety of structured prediction problems.

**Acknowledgments:** This research was partly funded by ARO grant W911NF-08-1-0242, AFRL contract FA8750-09-C-0181, NSF grant IIS-0803481, and ONR grant N00014-12-1-0312. The views and conclusions contained in this document are those of the authors and should not be interpreted as necessarily representing the official policies, either expressed or implied, of ARO, AFRL, NSF, ONR, or the United States Government.

## Footnotes

[1]A consistent SPN allows for MPE inference to reach the same indicator more than once in the same branching path

# References

[1] M. Amer and S. Todorovic. Sum-product networks for modeling activities with stochastic structure. *CVPR*, 2012.

[2] F. Bach and M.I. Jordan. Thin junction trees. *Advances in Neural Information Processing Systems*, 14:569–576, 2002.

[3] Y. Bengio. Learning deep architectures for AI. *Foundations and Trends in Machine Learning*, 2(1):1–127, 2009.

[4] L. Bo, K. Lai, X. Ren, and D. Fox. Object recognition with hierarchical kernel descriptors. In *Computer Vision and Pattern Recognition (CVPR), 2011 IEEE Conference on*, pages 1729–1736. IEEE, 2011.

[5] L. Bo, X. Ren, and D. Fox. Kernel descriptors for visual recognition. *Advances in Neural Information Processing Systems*, 2010.

[6] L. Bo, X. Ren, and D. Fox. Unsupervised feature learning for RGB-D based object recognition. *ISER*, 2012.

[7] C. Boutilier, N. Friedman, M. Goldszmidt, and D. Koller. Context-specific independence in bayesian networks. In *Proceedings of the Twelfth Conference on Uncertainty in Artificial Intelligence*, pages 115–123, 1996.

[8] M. Chavira and A. Darwiche. On probabilistic inference by weighted model counting. *Artificial Intelligence*, 172(6-7):772–799, 2008.

[9] A. Chechetka and C. Guestrin. Efficient principled learning of thin junction trees. In J.C. Platt, D. Koller, Y. Singer, and S. Roweis, editors, *Advances in Neural Information Processing Systems 20*. MIT Press, Cambridge, MA, 2008.

[10] A. Coates, H. Lee, and A.Y. Ng. An analysis of single-layer networks in unsupervised feature learning. In *aistats11*. Society for Artificial Intelligence and Statistics, 2011.

[11] A. Coates and A.Y. Ng. The importance of encoding versus training with sparse coding and vector quantization. In *International Conference on Machine Learning*, volume 8, page 10, 2011.

[12] A. Coates and A.Y. Ng. Selecting receptive fields in deep networks. NIPS, 2011.

[13] M. Collins. Discriminative training methods for hidden Markov models: Theory and experiments with perceptron algorithms. In *Proceedings of the 2002 Conference on Empirical Methods in Natural Language Processing*, pages 1–8, Philadelphia, PA, 2002. ACL.

[14] A. Darwiche. A differential approach to inference in Bayesian networks. *Journal of the ACM*, 50:280–305, 2003.

[15] A. Darwiche. *Modeling and Reasoning with Bayesian Networks*. Cambridge University Press, 2009.

[16] O. Delalleau and Y. Bengio. Shallow vs. deep sum-product networks. In *Proceedings of the 25th Conference on Neural Information Processing Systems*, 2011.

[17] A. P. Dempster, N. M. Laird, and D. B. Rubin. Maximum likelihood from incomplete data via the EM algorithm. *Journal of the Royal Statistical Society, Series B*, 39:1–38, 1977.

[18] P. Felzenszwalb, D. McAllester, and D. Ramanan. A discriminatively trained, multiscale, deformable part model. In *Computer Vision and Pattern Recognition, 2008. CVPR 2008. IEEE Conference on*, pages 1–8. Ieee, 2008.

[19] A. Hyvärinen and E. Oja. Independent component analysis: algorithms and applications. *Neural networks*, 13(4-5):411–430, 2000.

[20] Y. Jia, C. Huang, and T. Darrell. Beyond spatial pyramids: Receptive field learning for pooled image features. In *CVPR*, 2012.

[21] A. Kulesza, F. Pereira, et al. Structured learning with approximate inference. *Advances in Neural Information Processing Systems*, 20:785–792, 2007.

[22] J. Lafferty, A. McCallum, and F. Pereira. Conditional random fields: Probabilistic models for segmenting and labeling data. In *Proceedings of the Eighteenth International Conference on Machine Learning*, pages 282–289, Williamstown, MA, 2001. Morgan Kaufmann.

[23] H. Poon and P. Domingos. Sum-product networks: A new deep architecture. In *Proc. 12th Conf. on Uncertainty in Artificial Intelligence*, pages 337–346, 2011.

[24] M.A. Ranzato and G.E. Hinton. Modeling pixel means and covariances using factorized third-order Boltzmann machines. In *Computer Vision and Pattern Recognition (CVPR), 2010 IEEE Conference on*, pages 2551–2558. IEEE, 2010.

[25] J. Salojärvi, K. Puolamäki, and S. Kaski. Expectation maximization algorithms for conditional likelihoods. In *Proceedings of the 22nd international conference on Machine learning*, pages 752–759. ACM, 2005.

